# Learning in Zero-Sum Team Markov Games Using Factored Value Functions

**Michail G. Lagoudakis**
Department of Computer Science
Duke University
Durham, NC 27708
*mgl@cs.duke.edu*

**Ronald Parr**
Department of Computer Science
Duke University
Durham, NC 27708
*parr@cs.duke.edu*

## Abstract

We present a new method for learning good strategies in zero-sum Markov games in which each side is composed of multiple agents collaborating against an opposing team of agents. Our method requires full observability and communication during learning, but the learned policies can be executed in a distributed manner. The value function is represented as a factored linear architecture and its structure determines the necessary computational resources and communication bandwidth. This approach permits a tradeoff between simple representations with little or no communication between agents and complex, computationally intensive representations with extensive coordination between agents. Thus, we provide a principled means of using approximation to combat the exponential blowup in the joint action space of the participants. The approach is demonstrated with an example that shows the efficiency gains over naive enumeration.

## 1 Introduction

The Markov game framework has received increased attention as a rigorous model for defining and determining optimal behavior in multiagent systems. The zero-sum case, in which one side's gains come at the expense of the other's, is the simplest and best understood case[1]. Littman [7] demonstrated that reinforcement learning could be applied to Markov games, albeit at the expense of solving one linear program for each state visited during learning. This computational (and conceptual) burden is probably one factor behind the relative dearth of ambitious Markov game applications using reinforcement learning.

In recent work [6], we demonstrated that many previous theoretical results justifying the use of value function approximation to tackle large MDPs could be generalized to Markov games. We applied the LSPI reinforcement learning algorithm [5] with function approximation to a two-player soccer game and a router/server flow control problem and derived very good results. While the theoretical results [6] are general and apply to any reinforcement learning algorithm, we preferred to use LSPI because LSPI's efficient use of data meant that we solved fewer linear programs during learning.

Since soccer, routing, and many other natural applications of the Markov game framework tend to involve multiple participants it would be very useful to generalize recent advances in multiagent cooperative MDPs [2, 4] to Markov games. These methods use a factored value function architecture and determine the optimal action using a cost network [1] and a communication structure which is derived directly from the structure of the value function. LSPI has been successfuly combined with such methods; in empirical experiments, the number of state visits required to achieve good performance scaled linearly with the number of agents despite the exponential growth in the joint action space [4].

In this paper, we integrate these ideas and we present an algorithm for learning good strategies for a team of agents that plays against an opponent team. In such games, players within one team collaborate, whereas players in different teams compete. The key component of this work is a method for computing efficiently the best strategy for a team, given an approximate factored value function which is a linear combination of features defined over the state space and subsets of the joint action space for both sides. This method integrated within LSPI yields a computationally efficient learning algorithm.

## 2 Markov Games

A two-player zero-sum Markov game is defined as a 6-tuple $(\mathcal{S}, \mathcal{A}, \mathcal{O}, P, \mathcal{R}, \gamma)$, where: $\mathcal{S} = \{s_1, s_2, ..., s_n\}$ is a finite set of game states; $\mathcal{A} = \{a_1, a_2, ..., a_m\}$ and $\mathcal{O} = \{o_1, o_2, ..., o_l\}$ are finite sets of actions, one for each player; $P$ is a Markovian state transition model — $P(s, a, o, s')$ is the probability that $s'$ will be the next state of the game when the players take actions $a$ and $o$ respectively in state $s$; $\mathcal{R}$ is a reward (or cost) function — $\mathcal{R}(s, a, o)$ is the expected one-step reward for taking actions $a$ and $o$ in state $s$; and, $\gamma \in (0, 1]$ is the discount factor for future rewards. We will refer to the first player as the *maximizer* and the second player as the *minimizer*[2]. Note that if either player is permitted only a single action, the Markov game becomes an MDP for the other player.

A *policy* $\pi$ for a player in a Markov game is a mapping, $\pi : \mathcal{S} \to \Omega(\mathcal{A})$, which yields probability distributions over the maximizer's actions for each state in $\mathcal{S}$. Unlike MDPs, the optimal policy for a Markov game may be stochastic, i.e., it may define a *mixed* strategy for every state. By convention, for any policy $\pi$, $\pi(s)$ denotes the probability distribution over actions in state $s$ and $\pi(s, a)$ denotes the probability of action $a$ in state $s$.

The maximizer is interested in maximizing its expected, discounted return in the *minimax* sense, that is, assuming the worst case of an optimal minimizer. Since the underlying rewards are zero-sum, it is sufficient to view the minimizer as acting to minimize the maximizer's return. For any policy $\pi$, we can define $Q^\pi(s, a, o)$ as the expected total discounted reward of the maximizer when following policy $\pi$ after the players take actions $a$ and $o$ for the first step. The corresponding fixed point equation for $Q^\pi$ is:

$$Q^\pi(s, a, o) = \mathcal{R}(s, a, o) + \gamma \sum_{s' \in \mathcal{S}} P(s, a, o, s') \min_{o' \in \mathcal{O}} \sum_{a' \in \mathcal{A}} Q^\pi(s', a', o')\pi(s', a') .$$

Given any $Q$ function, the maximizer can choose actions so as to maximize its value:

$$V(s) = \max_{\pi'(s) \in \Omega(\mathcal{A})} \min_{o \in \mathcal{O}} \sum_{a \in \mathcal{A}} Q(s, a, o)\pi'(s, a) . \tag{1}$$

We will refer to the policy $\pi'$ chosen by Eq. (1) as the *minimax policy* with respect to $Q$.

This policy can be determined in any state $s$ by solving the following linear program:

$$
\begin{aligned}
\text{Maximize:} \quad & V(s) \\
\text{Subject to:} \quad & \forall a \in \mathcal{A},\ \pi'(s,a) \geq 0 \\
& \sum_{a \in \mathcal{A}} \pi'(s,a) = 1 \\
& \forall o \in \mathcal{O},\ V(s) \leq \sum_{a \in \mathcal{A}} Q(s,a,o)\pi'(s,a)\ .
\end{aligned}
$$

If $Q = Q^\pi$, the minimax policy is an improved policy compared to $\pi$. A *policy iteration* algorithm can be implemented for Markov games in a manner analogous to policy iteration for MDPs by fixing a policy $\pi_i$, solving for $Q^{\pi_i}$, choosing $\pi_{i+1}$ as the minimax policy with respect to $Q^{\pi_i}$ and iterating. This algorithm converges to the optimal minimax policy $\pi^*$.

## 3   Least Squares Policy Iteration (LSPI) for Markov Games

In practice, the state/action space is too large for an explicit representation of the $Q$ function. We consider the standard approach of approximating the $Q$ function as the linear combination of $k$ basis functions $\phi_j$ with weights $w_j$, that is $\widehat{Q}(s,a,o) = \phi(s,a,o)^{\mathsf{T}}w$. With this representation, the minimax policy $\pi$ for the maximizer is determined by

$$
\pi(s) = \underset{\pi(s)\,\in\Omega(\mathcal{A})}{\arg\max}\ \min_{o\in\mathcal{O}}\ \sum_{a\in\mathcal{A}} \pi(s,a)\phi(s,a,o)^{\mathsf{T}}w\ ,
$$

and can be computed by solving the following linear program

$$
\begin{aligned}
\text{Maximize:} \quad & V(s) \\
\text{Subject to:} \quad & \forall\, a \in \mathcal{A},\ \pi(s,a) \geq 0 \\
& \sum_{a \in \mathcal{A}} \pi(s,a) = 1 \\
& \forall\, o \in \mathcal{O},\ V(s) \leq \sum_{a \in \mathcal{A}} \pi(s,a)\phi(s,a,o)^{\mathsf{T}}w\ .
\end{aligned}
$$

We chose the LSPI algorithm to learn the weights $w$ of the approximate value function. Least-Squares Policy Iteration (LSPI) [5] is an approximate policy iteration algorithm that learns policies using a corpus of stored samples. LSPI applies also with minor modifications to Markov games [6]. In particular, at each iteration, LSPI evaluates the current policy using the stored samples and keeps the learned weights to represent implicitly the improved minimax policy for the next iteration by solving the linear program above. The modified update equations account for the minimizer's action and the distribution over next maximizer actions since the minimax policy is, in general, stochastic. More specifically, at each iteration LSPI maintains two matrices, $\widehat{\mathbf{A}}$ and $\widehat{b}$, which are updated as follows:

$$
\widehat{\mathbf{A}} \leftarrow \widehat{\mathbf{A}} + \phi(s,a,o)\Big(\phi(s,a,o) - \gamma \sum_{a'\in\mathcal{A}} \pi(s',a')\phi(s',a',o')\Big)^{\mathsf{T}},\qquad \widehat{b} \leftarrow \widehat{b} + \phi(s,a,o)r\ ,
$$

for any sample $(s,a,o,r,s')$. The policy $\pi'(s')$ for state $s'$ is computed using the linear program above. The action $o'$ is the minimizing opponent action in computing $\pi(s')$ and can be identified by the tight constraint on $V(s')$. The weight vector $w$ is computed at the end of each iteration as the solution to $\widehat{\mathbf{A}}w = \widehat{b}$. The key step in generalizing LSPI to team Markov games is finding efficient means to perform these operations despite the exponentially large joint action space.

## 4   Least Squares Policy Iteration for Team Markov Games

A *team Markov game* is a Markov game where a team of $N$ maximizers is playing against a team of $M$ minimizers. Maximizer $i$ chooses actions from $\mathcal{A}_i$, so the team chooses

actions $\bar{a} = (a_1, a_2, ..., a_N)$ from $\bar{\mathcal{A}} = \mathcal{A}_1 \times \mathcal{A}_2 \times ... \times \mathcal{A}_N$, where $a_i \in \mathcal{A}_i$. Minimizer $i$ chooses actions from $\mathcal{O}_i$, so the minimizer team chooses actions $\bar{o} = (o_1, o_2, ..., o_M)$ from $\bar{\mathcal{O}} = \mathcal{O}_1 \times \mathcal{O}_2 \times ... \times \mathcal{O}_M$, where $o_i \in \mathcal{O}_i$. Consider now an approximate value function $\widehat{Q}(s, \bar{a}, \bar{o})$. The minimax policy $\pi$ for the maximizer team in any given state $s$ can be computed (naively) by solving the following linear program:

$$
\begin{aligned}
\text{Maximize:} \quad & V(s) \\
\text{Subject to:} \quad & \forall\, \bar{a} \in \bar{\mathcal{A}}, \ \pi(s, \bar{a}) \geq 0 \\
& \sum_{\bar{a} \in \bar{\mathcal{A}}} \pi(s, \bar{a}) = 1 \\
& \forall\, \bar{o} \in \bar{\mathcal{O}}, \ V(s) \leq \sum_{\bar{a} \in \bar{\mathcal{A}}} \pi(s, \bar{a})\widehat{Q}(s, \bar{a}, \bar{o}) \ .
\end{aligned}
$$

Since $|\bar{\mathcal{A}}|$ is exponential in $N$ and $|\bar{\mathcal{O}}|$ is exponential in $M$, the linear program above has an exponential number of variables and constraints and would be intractable to solve, unless we make certain assumptions about $\widehat{Q}$. We assume a *factored* approximation [2] of the $Q$ function, given as a linear combination of $k$ *localized* basis functions. Each basis function can be thought of as an individual player's perception of the environment, so each $\phi_j$ need not depend upon every feature of the state or the actions taken by every player in the game. In particular, we assume that each $\phi_j$ depends only on the actions of a small subset of maximizers $A_j$ and minimizers $O_j$, that is, $\phi_j = \phi_j(s, \bar{a}_j, \bar{o}_j)$, where $\bar{a}_j \in \bar{\mathcal{A}}_j$ and $\bar{o}_j \in \bar{\mathcal{O}}_j$ ($\bar{\mathcal{A}}_j$ is the joint action space of the plyers in $A_j$ and $\bar{\mathcal{O}}_j$ is the joint action space of the players in $O_j$). For example, if $\phi_4$ depends only on the actions of maximizers $\{4, 5, 8\}$, and the actions of minimizers $\{3, 2, 7\}$, then $\bar{a}_4 \in \mathcal{A}_4 \times \mathcal{A}_5 \times \mathcal{A}_8$ and $\bar{o}_4 \in \mathcal{O}_3 \times \mathcal{O}_2 \times \mathcal{O}_7$. Under this locality assumption, the approximate (factored) value function is

$$
\widehat{Q}(s, \bar{a}, \bar{o}) = \sum_{j=1}^{k} \phi_j(s, \bar{a}_j, \bar{o}_j) w_j \ ,
$$

where the assignments to the $\bar{a}_j$'s and $\bar{o}_j$'s are consistent with $\bar{a}$ and $\bar{o}$. Given this form of the value function the linear program can be simplified significantly. We look at the constraints for the value of the state first:

$$
\begin{aligned}
V(s) \ &\leq \ \sum_{\bar{a} \in \bar{\mathcal{A}}} \pi(s, \bar{a}) \sum_{j=1}^{k} \phi_j(s, \bar{a}_j, \bar{o}_j) w_j \\
V(s) \ &\leq \ \sum_{j=1}^{k} \sum_{\bar{a} \in \bar{\mathcal{A}}} \pi(s, \bar{a}) \phi_j(s, \bar{a}_j, \bar{o}_j) w_j \\
V(s) \ &\leq \ \sum_{j=1}^{k} \sum_{\bar{a}_j \in \bar{\mathcal{A}}_j} \sum_{\bar{a}' \in \bar{\mathcal{A}} \setminus \bar{\mathcal{A}}_j} \pi(s, \bar{a}) \phi_j(s, \bar{a}_j, \bar{o}_j) w_j \\
V(s) \ &\leq \ \sum_{j=1}^{k} w_j \sum_{\bar{a}_j \in \bar{\mathcal{A}}_j} \phi_j(s, \bar{a}_j, \bar{o}_j) \sum_{\bar{a}' \in \bar{\mathcal{A}} \setminus \bar{\mathcal{A}}_j} \pi(s, \bar{a}) \\
V(s) \ &\leq \ \sum_{j=1}^{k} w_j \sum_{\bar{a}_j \in \bar{\mathcal{A}}_j} \phi_j(s, \bar{a}_j, \bar{o}_j) \pi_j(s, \bar{a}_j) \ ,
\end{aligned}
$$

where each $\pi_j(s, \bar{a}_j)$ defines a probability distribution over the actions of the players that appear in $\phi_j$. From the last expression, it is clear that we can use $\pi_j(s, \bar{a}_j)$ as the variables of the linear program. The number of these variables will typically be much smaller than the number of variables $\pi(s, \bar{a})$, depending on the size of the $A_j$'s. However, we must add constraints to ensure that the local probability distributions $\pi_j(s)$ are consistent with a global distribution over the entire joint action space $\bar{\mathcal{A}}$. The first set of constraints are the

standard ones for any probability distribution:

$$\forall\, j = 1, ..., k \quad : \quad \sum_{\bar{a}_j \in \bar{\mathcal{A}}_j} \pi_j(s, \bar{a}_j) = 1$$

$$\forall\, j = 1, ..., k \quad : \quad \forall\, \bar{a}_j \in \bar{\mathcal{A}}_j,\ \pi_j(s, \bar{a}_j) \geq 0 \ .$$

For consistency, we must ensure that all marginals over common variables are identical:

$$\forall\, 1 \leq j < h \leq k \quad : \quad \forall\, \bar{a}' \in \bar{\mathcal{A}}_j \cap \bar{\mathcal{A}}_h, \sum_{\bar{a}'_j \in \bar{\mathcal{A}}_j \setminus \bar{\mathcal{A}}_h} \pi_j(s, \bar{a}_j) = \sum_{\bar{a}'_h \in \bar{\mathcal{A}}_h \setminus \bar{\mathcal{A}}_j} \pi_h(s, \bar{a}_h) \ .$$

These constraints are sufficient if the running intersection property is satisfied by the $\pi_j(s)$'s [3]. If not, it is possible that the resulting $\pi_j(s)$'s will not be consistent with any global distribution even though they are locally consistent. However, the running intersection property can be enforced by introducing certain additional local distributions in the set of $\pi_j(s)$'s. This can be achieved using a variable elimination procedure.

First, we establish an elimination order for the maximizers and we let $\mathcal{H}_1$ be the set of all $\pi_j(s)$'s and $\mathcal{L} = \varnothing$. At each step $i$, some agent $i$ is eliminated and we let $\mathcal{E}_i$ be the set of all distributions in $\mathcal{H}_i$ that involve the actions of agent $i$ or have empty domain. We then create a new distribution $\omega_i$ over the actions of all agents that appear in $\mathcal{E}_i$ and we place $\omega_i$ in $\mathcal{L}$. We then create $\omega'_i$ defined as the distribution over the actions of all agents that appear in $\omega_i$ except agent $i$. Next, we update $\mathcal{H}_{i+1} = \mathcal{H}_i \cup \{\omega'_i\} - \mathcal{E}_i$ and repeat until all agents have been eliminated. Note that $\mathcal{H}_N$ will necessarily be empty and $\mathcal{L}$ will contain at most $N$ new local probability distributions. We can manipulate the elimination order in an attempt to keep the distributions in $\mathcal{L}$ small (local), however their size will be exponential in the induced tree width. As with Bayes nets, the existence and hardness of discovering efficient elimination orderings will depend upon the topology. The set $\mathcal{H}_1 \cup \mathcal{L}$ of local probability distributions satisfies the running intersection property and so we can proceed with this set instead of the original set of $\pi_j(s)$'s and apply the constraints listed above. Even though we are only interested in the $\pi_j(s)$'s, the existence of the additional distributions in the linear program will ensure that the $\pi_j(s)$'s will be globally consistent.

The number of constraints needed for the local probability distributions is much smaller than the original number of constraints. In summary, the new linear program will be:

Maximize: $V(s)$
Subject to: $\forall\, j = 1, ..., k\ :\ \forall\, \bar{a}_j \in \bar{\mathcal{A}}_j,\ \pi_j(s, \bar{a}_j) \geq 0$

$$\forall\, j = 1, ..., k\ :\ \sum_{\bar{a}_j \in \bar{\mathcal{A}}_j} \pi_j(s, \bar{a}_j) = 1$$

$$\forall\, 1 \leq j < h \leq k\ :\ \forall\, \bar{a}' \in \bar{\mathcal{A}}_j \cap \bar{\mathcal{A}}_h, \sum_{\bar{a}'_j \in \bar{\mathcal{A}}_j \setminus \bar{\mathcal{A}}_h} \pi_j(s, \bar{a}_j) = \sum_{\bar{a}'_h \in \bar{\mathcal{A}}_h \setminus \bar{\mathcal{A}}_j} \pi_h(s, \bar{a}_h)$$

$$\forall\, \bar{o} \in \bar{\mathcal{O}},\ V(s) \leq \sum_{j=1}^{k} w_j \sum_{\bar{a}_j \in \bar{\mathcal{A}}_j} \phi_j(s, \bar{a}_j, \bar{o}_j) \pi_j(s, \bar{a}_j) \ .$$

At this point we have eliminated the exponential dependency from the number of variables and partially from the number of constraints. The last set of (exponentially many) constraints can be replaced by a single non-linear constraint:

$$V(s) \leq \min_{\bar{o} \in \bar{\mathcal{O}}} \sum_{j=1}^{k} w_j \sum_{\bar{a}_j \in \bar{\mathcal{A}}_j} \phi_j(s, \bar{a}_j, \bar{o}_j) \pi_j(s, \bar{a}_j) \ .$$

We now show how this non-linear constraint can be turned into a number of linear constraints which is not exponential in $M$ in general. The main idea is to embed a cost network inside the linear program [2]. In particular, we define an elimination order for the $o_i$'s in $\bar{o}$

and, for each $o_i$ in turn, we push the $\min$ operator for just $o_i$ as far inside the summation as possible, keeping only terms that have some dependency on $o_i$ or no dependency on any of the opponent team actions. We replace this smaller $\min$ expression over $o_i$ with a new function $f_i$ (represent by a set of new variables in the linear program) that depends on the other opponent actions that appear in this $\min$ expression. Finally, we introduce a set of linear constraints for the value of $f_i$ that express the fact that $f_i$ is the minimum of the eliminated expression in all cases. We repeat this elimination process until all $o_i$'s and therefore all $\min$ operators are eliminated.

More formally, at step $i$ of the elimination, let $\mathcal{B}_i$ be the set of basis functions that have not been eliminated up to that point and $\mathcal{F}_i$ be the set of the new functions that have not been eliminated yet. For simplicity, we assume that the elimination order is $o_1, o_2, ..., o_M$ (in practice the elimination order needs to be chosen carefully in advance since a poor elimination ordering could have serious adverse effects on efficiency). At the very beginning of the elimination process, $\mathcal{B}_1 = \{\phi_1, \phi_2, ..., \phi_k\}$ and $\mathcal{F}_1$ is empty. When eliminating $o_i$ at step $i$, define $\mathcal{E}_i \subseteq \mathcal{B}_i \cup \mathcal{F}_i$ to be those functions that contain $o_i$ in their domain or have no dependency on any opponent action. We generate a new function $f_i(\bar{\bar{o}}_i)$ that depends on all the opponent actions that appear in $\mathcal{E}_i$ excluding $o_i$:

$$f_i(\bar{\bar{o}}_i) = \min_{o_i \in \mathcal{O}_i} \left\{ \sum_{\phi_j \in \mathcal{E}_i} w_j \sum_{\bar{a}_j \in \bar{\mathcal{A}}_j} \phi_j(s, \bar{a}_j, \bar{o}_j) \pi_j(s, \bar{a}_j) + \sum_{f_k \in \mathcal{E}_i} f_k(\bar{\bar{o}}_k) \right\} \ .$$

We introduce a new variable in the linear program for each possible setting of the domain $\bar{\bar{o}}_i$ of the new function $f_i(\bar{\bar{o}}_i)$. We also introduce a set of constraints for these variables:

$$\forall \, o_i \in \mathcal{O}_i, \ \forall \, \bar{\bar{o}}_i \ : \ f_i(\bar{\bar{o}}_i) \leq \sum_{\phi_j \in \mathcal{E}_i} w_j \sum_{\bar{a}_j \in \bar{\mathcal{A}}_j} \phi_j(s, \bar{a}_j, \bar{o}_j) \pi_j(s, \bar{a}_j) + \sum_{f_k \in \mathcal{E}_i} f_k(\bar{\bar{o}}_k)$$

These constraints ensure that the new function is the minimum over the possible choices for $o_i$. Now, we define $\mathcal{B}_{i+1} = \mathcal{B}_i - \mathcal{E}_i$ and $\mathcal{F}_{i+1} = \mathcal{F}_i - \mathcal{E}_i + \{f_i\}$ and we continue with the elimination of action $o_{i+1}$. Notice that $o_i$ does not appear anywhere in $\mathcal{B}_{i+1}$ or $\mathcal{F}_{i+1}$. Notice also that $f_M$ will necessarily have an empty domain and it is exactly the value of the state, $f_M = V(s)$. Summarizing everything, the reduced linear program is

Maximize:   $f_M$
Subject to:   $\forall \, j = 1, ..., k \ : \ \forall \, \bar{a}_j \in \bar{\mathcal{A}}_j, \ \pi_j(s, \bar{a}_j) \geq 0$

$$\forall \, j = 1, ..., k \ : \ \sum_{\bar{a}_j \in \bar{\mathcal{A}}_j} \pi_j(s, \bar{a}_j) = 1$$

$$\forall \, 1 \leq j < h \leq k \ : \forall \, \bar{a}' \in \bar{\mathcal{A}}_j \cap \bar{\mathcal{A}}_h, \ \sum_{\bar{a}'_j \in \bar{\mathcal{A}}_j \setminus \bar{A}_h} \pi_j(s, \bar{a}_j) = \sum_{\bar{a}'_h \in \bar{\mathcal{A}}_h \setminus \bar{A}_j} \pi_h(s, \bar{a}_h)$$

$$\forall \, i, \ \forall \, o_i, \ \forall \, \bar{\bar{o}}_i \ : \ f_i(\bar{\bar{o}}_i) \leq \sum_{\phi_j \in \mathcal{E}_i} w_j \sum_{\bar{a}_j \in \bar{\mathcal{A}}_j} \phi_j(s, \bar{a}_j, \bar{o}_j) \pi_j(s, \bar{a}_j) + \sum_{f_k \in \mathcal{E}_i} f_k(\bar{\bar{o}}_k)$$

Notice that the exponential dependency in $N$ and $M$ has been eliminated. The total number of variables and/or constraints is now exponentially dependent only on the number of players that appear together as a group in any of the basis functions or the intermediate functions and distributions. It should be emphasized that this reduced linear program solves the same problem as the naive linear program and yields the same solution (albeit in a factored form).

To complete the learning algorithm, the update equations of LSPI must also be modified. For any sample $(s, \bar{a}, \bar{o}, r, s')$, the naive form would be

$$\hat{\mathbf{A}} \leftarrow \hat{\mathbf{A}} + \phi(s, \bar{a}, \bar{o}) \Big( \phi(s, \bar{a}, \bar{o}) - \gamma \sum_{\bar{a}' \in \bar{\mathcal{A}}} \pi(s', \bar{a}') \phi(s', \bar{a}', \bar{o}') \Big)^{\mathsf{T}}, \quad \hat{b} \leftarrow \hat{b} + \phi(s, \bar{a}, \bar{o}) r \ .$$

The action $\bar{o}'$ is the minimizing opponent's action in computing $\pi(s')$. Unfortunately, the number of terms in the summation within the first update equation is exponential in

$N$. However, the vector $\phi(s,\bar{a},\bar{o}) - \gamma \sum_{\bar{a}' \in \bar{\mathcal{A}}} \pi(s',\bar{a}')\phi(s',\bar{a}',\bar{o}')$ can be computed on a component-by-component basis avoiding this exponential blowup. In particular, the $j$-th component is:

$$\phi_j(s,\bar{a}_j,\bar{o}) - \gamma \sum_{\bar{a}' \in \bar{\mathcal{A}}} \pi(s',\bar{a}')\phi_j(s',\bar{a}'_j,\bar{o}')$$

$$= \phi_j(s,\bar{a},\bar{o}) - \gamma \sum_{\bar{a}'_j \in \bar{\mathcal{A}}_j} \sum_{\bar{a}''_j \in \bar{\mathcal{A}} \setminus \bar{\mathcal{A}}_j} \pi(s',\bar{a}')\phi_j(s',\bar{a}'_j,\bar{o}')$$

$$= \phi_j(s,\bar{a},\bar{o}) - \gamma \sum_{\bar{a}'_j \in \bar{\mathcal{A}}_j} \phi_j(s',\bar{a}'_j,\bar{o}') \sum_{\bar{a}''_j \in \bar{\mathcal{A}} \setminus \bar{\mathcal{A}}_j} \pi(s',\bar{a}')$$

$$= \phi_j(s,\bar{a},\bar{o}) - \gamma \sum_{\bar{a}'_j \in \bar{\mathcal{A}}_j} \phi_j(s',\bar{a}'_j,\bar{o}')\pi_j(s',\bar{a}'_j) \ ,$$

which can be easily computed without exponential enumeration.

A related question is how to find $\bar{o}'$, the minimizing opponent's joint action in computing $\pi(s')$. This can be done after the linear program is solved by going through the $f_i$'s in reverse order (compared to the elimination order) and finding the choice for $o_i$ that imposes a tight constraint on $f_i(\bar{\bar{o}}_i)$ conditioned on the minimizing choice for $\bar{\bar{o}}_i$ that has been found so far. The only complication is that the linear program has no incentive to maximize $f_i(\bar{\bar{o}}_i)$ unless it contributes to maximizing the final value. Thus, a constraint that appears to be tight may not correspond to the actual minimizing choice. The solution to this is to do a forward pass first (according to the elimination order) marking the $f_i(\bar{\bar{o}}_i)$'s that really come from tight constraints. Then, the backward pass described above will find the true minimizing choices by using only the marked $f_i(\bar{\bar{o}}_i)$'s.

The last question is how to sample an action $\bar{a}$ from the global distribution defined by the smaller distributions. We begin with all actions uninstantiated and we go through all $\pi_j(s)$'s. For each $j$, we marginalize out the instantiated actions (if any) from $\pi_j(s)$ to generate the conditional probability and then we sample jointly the actions that remain in the distribution. We repeat with the next $j$ until all actions are instantiated. Notice that this operation can be performed in a distributed manner, that is, at execution time only agents whose actions appear in the same $\pi_j(s)$ need to communicate to sample actions jointly. This communication structure is directly derived from the structure of the basis functions.

## 5 An Example

The algorithm has been implemented and is currently being tested on a large flow control problem with multiple routers and servers. Since experimental results are still in progress, we demonstrate the efficiency gained over exponential enumeration with an example. Consider a problem with $N = 5$ maximizers and $M = 4$ minimizers. Assume also that each maximizer or minimizer has $5$ actions to choose from. The naive solution would require solving a linear program with 3126 variables and 3751 constraints for any representation of the value function. Consider now the following factored value function:

$$\widehat{Q}(s,\bar{a},\bar{o}) = \phi_1(s,a_1,a_2,o_1,o_2)w_1 \quad + \quad \phi_2(s,a_1,a_3,o_1,o_3)w_2 +$$
$$\phi_3(s,a_2,a_4,o_3)w_3 \quad + \quad \phi_4(s,a_3,a_5,o_4)w_4 + \phi_5(s,a_1,o_3,o_4)w_5 \ .$$

These basis functions satisfy the running intersection property (there is no cycle of length longer than 3), so there is no need for additional probability distributions. Using the elimination order $\{o_4, o_3, o_1, o_2\}$ for the cost network, the reduced linear program contains only 121 variables and 215 constraints (we present only the 80 constraints on the value of the state that demonstrate the variable elimination procedure, omitting the common constrains for validity and consistency of the local probability distributions):

$$\text{Maximize:} \quad f_2 \qquad \text{Subject to:}$$

$$\forall \, o_4 \in \mathcal{O}_4, \; \forall \, o_3 \in \mathcal{O}_3, \; f_4(o_3) \leq \sum_{(a_3, a_5) \in \mathcal{A}_3 \times \mathcal{A}_5} w_4 \phi_4(s, a_3, a_5, o_4) \pi_4(s, a_3, a_5) \quad +$$

$$\sum_{a_1 \in \mathcal{A}_1} w_5 \phi_5(s, a_1, o_3, o_4) \pi_5(s, a_1)$$

$$\forall \, o_3 \in \mathcal{O}_3, \; \forall \, o_1 \in \mathcal{O}_1, \; f_3(o_1) \leq \sum_{(a_1, a_3) \in \mathcal{A}_1 \times \mathcal{A}_3} w_2 \phi_2(s, a_1, a_3, o_1, o_3) \pi_2(s, a_1, a_3) \quad +$$

$$\sum_{(a_2, a_4) \in \mathcal{A}_2 \times \mathcal{A}_4} w_3 \phi_3(s, a_2, a_4, o_3) \pi_3(s, a_2, a_4) \quad + \quad f_4(o_3)$$

$$\forall \, o_1 \in \mathcal{O}_1, \; \forall \, o_2 \in \mathcal{O}_2, \; f_1(o_2) \leq \sum_{(a_1, a_2) \in \mathcal{A}_1 \times \mathcal{A}_2} w_1 \phi_1(s, a_1, a_2, o_1, o_2) \pi_1(s, a_1, a_2) + f_3(o_1)$$

$$\forall \, o_2 \in \mathcal{O}_2, \; f_2 \leq f_1(o_2)$$

## 6   Conclusion

We have presented a principled approach to the problem of solving large team Markov games that builds on recent advances in value function approximation for Markov games and multiagent coordination in reinforcement learning for MDPs. Our approach permits a tradeoff between simple architectures with limited representational capability and sparse communication and complex architectures with rich representations and more complex coordination structure. It is our belief that the algorithm presented in this paper can be used successfully in real-world, large-scale domains where the available knowledge about the underlying structure can be exploited to derive powerful and sufficient factored representations.

## Acknowledgments

This work was supported by NSF grant 0209088. We would also like to thank Carlos Guestrin for helpful discussions.

## Footnotes

[1]The term *Markov game* in this paper refers to the zero-sum case unless stated otherwise.

[2]Because of the duality, we adopt the maximizer's point of view for presentation.

## References

[1] R. Dechter. Bucket elimination: A unifying framework for reasoning. *Artificial Intelligence*, 113(1–2):41–85, 1999.

[2] Carlos Guestrin, Daphne Koller, and Ronald Parr. Multiagent planning with factored MDPs. In *Proceeding of the 14th Neural Information Processing Systems (NIPS-14)*, pages 1523–1530, Vancouver, Canada, December 2001.

[3] Carlos Guestrin, Daphne Koller, and Ronald Parr. Solving factored POMDPs with linear value functions. In *IJCAI-01 workshop on Planning under Uncertainty and Incomplete Information*, 2001.

[4] Carlos Guestrin, Michail G. Lagoudakis, and Ronald Parr. Coordinated reinforcement learning. In *Proceedings of the 19th International Conference on Machine Learning (ICML-02)*, pages 227–234, Sydney, Australia, July 2002.

[5] Michail Lagoudakis and Ronald Parr. Model free least squares policy iteration. In *Proceedings of the 14th Neural Information Processing Systems (NIPS-14)*, pages 1547–1554, Vancouver, Canada, December 2001.

[6] Michail Lagoudakis and Ronald Parr. Value function approximation in zero sum Markov games. In *Proceedings of the 18th Conference on Uncertainty in Artificial Intelligence (UAI 2002)*, pages 283–292, Edmonton, Canada, 2002.

[7] Michael L. Littman. Markov games as a framework for multi-agent reinforcement learning. In *Proceedings of the 11th International Conference on Machine Learning (ICML-94)*, pages 157–163, San Francisco, CA, 1994. Morgan Kaufmann.
